# Visual Question Answering with Question Representation Update (QRU)

**Ruiyu Li**          **Jiaya Jia**
The Chinese University of Hong Kong
{ryli,leojia}@cse.cuhk.edu.hk

## Abstract

Our method aims at reasoning over natural language questions and visual images. Given a natural language question about an image, our model updates the question representation iteratively by selecting image regions relevant to the query and learns to give the correct answer. Our model contains several reasoning layers, exploiting complex visual relations in the visual question answering (VQA) task. The proposed network is end-to-end trainable through back-propagation, where its weights are initialized using pre-trained convolutional neural network (CNN) and gated recurrent unit (GRU). Our method is evaluated on challenging datasets of COCO-QA [19] and VQA [2] and yields state-of-the-art performance.

## 1  Introduction

Visual question answering (VQA) is a new research direction as intersection of computer vision and natural language processing. Developing stable systems for VQA attracts increasing interests in multiple communities. Possible applications include bidirectional image-sentence retrieval, human computer interaction, blind person assistance, etc. It is now still a difficult problem due to many challenges in visual object recognition and grounding, natural language representation, and common sense reasoning.

Most recently proposed VQA models are based on image captioning [10, 24, 28]. These methods have been advanced by the great success of deep learning on building language models [23], image classification [12] and on visual object detection [6]. Compared with image captioning, where a plausible description is produced for a given image, VQA requires algorithms to give the correct answer to a specific human-raised question regarding the content of a given image. It is a more complex research problem since the method is required to answer different types of questions. An example related to image content is "`What is the color of the dog?`". There are also questions requiring extra knowledge or commonsense reasoning, such as "`Does it appear to be rainy?`".

Properly modeling questions is essential for solving the VQA problem. A commonly employed strategy is to use a CNN or an RNN to extract semantic vectors. The general issue is that the resulting question representation lacks detailed information from the given image, which however is vital for understanding visual content. We take the question and image in Figure 1 as an example. To answer the original question "`What is sitting amongst things have been abandoned?`", one needs to know the target object location. Thus the question can be more specific as "`What is discarded on the side of a building near an old book shelf?`".

In this paper, we propose a neural network based reasoning model that is able to update the question representation iteratively by inferring image information. With this new system, it is now possible to make questions more specific than the original ones focusing on important image information automatically. Our approach is based on neural reasoner [18], which has recently shown remarkable

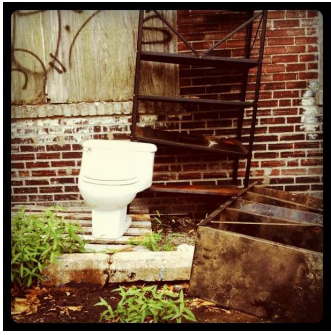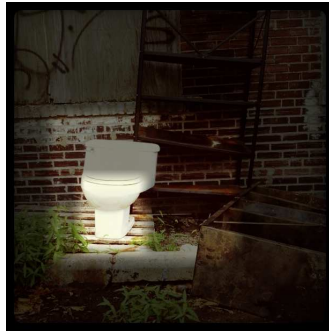

**Question:** What is sitting amongst things have been abandoned?
**Answer:** Toilet.

**Before:** What sits in the room that appears to be partially abandoned?

**Updated:** What is discarded on the side of a building near an old book shelf?

(a)             (b)

Figure 1: The questions asked by human can be ambiguous given an image containing various objects. The **Before** and **Updated** questions are the most similar ones based on the cosine similarity to the original **Question** before and after applying our algorithm to update representation. (b) shows the attention masks generated by our model.

success in text question answering tasks. Neural reasoner updates the question by interacting it with supporting facts through multiple reasoning layers. We note applying this model to VQA is nontrivial since the facts are in the form of an image. Thus image region information is extracted in our model. To determine the relevance between question and each image region, we employ the attention mechanism to generate the attention distribution over regions of the image. Our contributions are as follows.

- We present a reasoning network to iteratively update the question representation after each time the question interacts with image content.
- Our model utilizes object proposals to obtain candidate image regions and has the ability to focus on image regions relevant to the question.

We evaluate and compare the performance of our model on two challenging VQA datasets – i.e., COCO-QA [19] and VQA [2]. Experiments demonstrate the ability of our model to infer image regions relevant to the question.

## 2 Related Work

Research on visual question answering is mostly driven by text question answering and image captioning methods. In natural language processing, question answering is a well-studied problem. In [22], an end-to-end memory network was used with a recurrent attention model over a large external memory. Compared with the original memory network, it has less supervision and shows comparable results on the QA task. The neural reasoning system proposed in [18], named neural reasoner, can utilize multiple supporting facts and find an answer. Decent performance was achieved on positional reasoning and path finding QA tasks.

VQA is closely related to image captioning [10, 24, 28, 5]. In [5], a set of likely words are detected in several regions of the image and are combined together using a language model to generate image description. In [10], a structured max-margin objective was used for deep neural networks. It learns to embed both visual and language data into a common multi-modal space. Vinyals et al. [24] extracted high-level image feature vectors from CNN and took them as the first input to the recurrent network to generate caption. Xu et al. [28] integrated visual attention in the recurrent network. The proposed algorithm predicts one word at a time by looking at local image regions relevant to the currently generated word.

Malinowski et al. [15] first introduced a solution addressing the VQA problem. It combines natural language processing with semantic segmentation in a Bayesian framework for automatic question answering. Since it, several neural network based models [16, 19, 2] were proposed to solve the VQA problem. These models use CNN to extract image features and recurrent neural networks to embed questions. The embedded image and question features are then fused by concatenation [16]

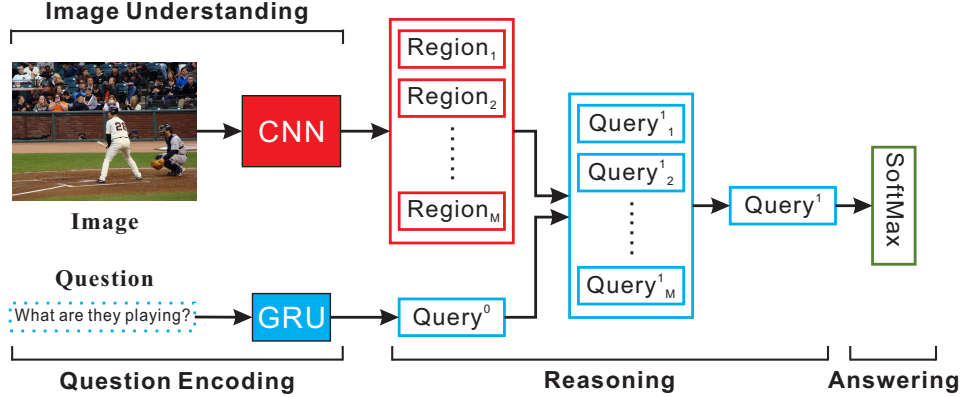

Figure 2: The overall architecture of our model with single reasoning layer for VQA.

or element-wise addition [29] to predict answers. Recently several models integrated the attention mechanism [29, 27, 3, 20] and showed the ability of their networks to focus on image regions related to the question.

There also exist other approaches for VQA. For example, Xiong et al. [26] proposed an improved dynamic memory network to fuse the question and image region representations using bi-directional GRU. The algorithm of [1] learns to compose a network from a collection of composable modules. Ma et al. [14] made use of CNN and proposed a model with three CNNs to capture information of the image, question and multi-modal representation.

## 3 Our Model

The overall architecture of our model is illustrated in Figure 2. The model is derived from the neural reasoner [18], which is able to update the representation of question recursively by inferring over multiple supporting facts. Our model yet contains a few inherently different components. Since VQA involves only one question and one image each time instead of a set of facts, we use object proposal to obtain candidate image regions serving as the facts in our model. Moreover, in the pooling step, we employ an attention mechanism to determine the relevance between representation of original questions and updated ones. Our network consists of four major components – i.e., image understanding, question encoding, reasoning and answering layers.

### 3.1 Image Understanding Layer

The image understanding layer is designed for modeling image content into semantic vectors. We build this layer upon the VGG model with 19 weight layers [21]. It is pre-trained on ImageNet [4]. The network has sixteen convolutional layers and five max-pooling layers of kernel size $2 \times 2$ with stride 2, followed by two fully-connected layers with 4,096 neurons.

Using a global representation of the image may fail to capture all necessary information for answering the question involving multiple objects and spatial configuration. Moreover, since most of the questions are related to objects [19, 2], we utilize object proposal generator to produce a set of candidate regions that are most likely to be an object. For each image, we choose candidate regions by extracting the top 19 detected edge boxes [31]. We choose intersection over union (IoU) value 0.3 when performing non-maximum suppression, which is a common setting in object detection.

Additionally, the whole image region is added to capture the global information in the image understanding layer, resulting in 20 candidate regions per image. We extract features from each candidate region through the above mentioned CNN, bringing a dimension of 4,096 image region features. The extracted features, however, lack spatial information for object location. To remedy this issue, we follow the method of [8] to include an 8D representation

$$[x_{min}, y_{min}, x_{max}, y_{max}, x_{center}, y_{center}, w_{box}, h_{box}],$$

where $w_{box}$ and $h_{box}$ are the width and height of the image region. We set the image center as the origin. The coordinates are normalized to range from $-1$ to 1. Then each image region is represented as a 4104D feature denoted as $f_i$ where $i \in [1, 20]$. For modeling convenience, we use a single layer perceptron to transform the image representation into a common latent space shared with the question feature

$$v_i = \phi(W_{vf} * f_i + b_{vf}), \tag{1}$$

where $\phi$ is the rectified activation function $\phi(x) = max(0, x)$.

### 3.2 Question Encoding Layer

To encode the natural language question, we resort to the recurrent neural network, which has demonstrated great success on sentence embedding. The question encoding layer is composed of a word embedding layer and GRU cells. Given a question $w = [w_1, ..., w_T]$, where $w_t$ is the $t$th word in the question and $T$ is the length of the question, we first embed each word $w_t$ to a vector space $x_t$ with an embedding matrix $x_t = W_e w_t$. Then for each time step, we feed $x_t$ into GRU sequentially. At each step, the GRU takes one input vector $x_t$, and updates and outputs a hidden state $h_t$. The final hidden state $h_T$ is considered as the question representation. We also embed it into the common latent space same as image embedding through a single layer perceptron

$$q = \phi(W_{qh} * h_T + b_{qh}). \tag{2}$$

We utilize the pre-trained network with skip-thought vectors model [11] designed for general sentence embedding to initialize our question encoding layer as used in [17]. Note that the skip-thought vectors model is trained in an unsupervised manner on large language corpus. By fine-tuning the GRU, we transfer knowledge from natural language corpus to the VQA problem.

### 3.3 Reasoning Layer

The reasoning layer includes question-image interaction and weighted pooling.

**Question-Image Interaction**    Given that multilayer perceptron (MLP) has the ability to determine the relationship between two input sentences according to supervision [7, 18]. We examine image region features and question representation to acquire a good understanding of the question. In a memory network [22], these image region features are akin to the input memory representation, which can be retrieved for multiple times according to the question.

There are a total of $L$ reasoning layers. In the $l$th reasoning layer, the $i$th interaction happens between $q^{l-1}$ and $v_i$ through an MLP, resulting in updated question representation $q_i^l$ as

$$q_i^l = MLP_l(q^{l-1}, v_i; \theta_l), \tag{3}$$

with $\theta_l$ being the model parameter of interaction at the $l$th reasoning layer. In the simplest case with one single layer in $MLP_l$, the updating process is given by

$$q_i^l = \phi(W_l * (q^{l-1} \otimes v_i) + b_l), \tag{4}$$

where $\otimes$ indicates element-wise multiplication, which performs better in our experiments than other strategies, e.g., concatenation and element-wise addition.

Generally speaking, $q_i^l$ contains update of network focus towards answering the question after its interaction with image feature $v_i$. This property is important for the reasoning process [18].

**Weighted Pooling**    Pooling aims to fuse components of the question after its interaction with all image features to update representation. Two common strategies for pooling are max and mean pooling. However, when answering a specific question, it is often the case the correct answer is only related to particular image regions. Therefore, using max pooling may lead to unsatisfying results since questions may involve interaction between human and object, while mean pooling may also cause inferior performance due to noise introduced by regions irrelevant to the question.

To determine the relevance between question and each image region, we resort to the attention mechanism used in [28] to generate the attention distribution over image regions. For each updated

question $q_i^l$ after interaction with the $i$th image region, it is chosen close to the original question representation $q^{l-1}$. Hence, the attention weights take the following forms.

$$C_i = tanh(W_A * q_i^l \oplus (W_B * q^{l-1} + b_B)),$$
$$P = softmax(W_P * C + b_P), \tag{5}$$

where $C$ is a matrix and its $i$th column is $C_i$. $P \in \mathbb{R}^M$ is a $M$ dimensional vector representing the attention weights. $M$ is the number of image regions, set to 20. Based on the attention distribution, we calculate weighted average of $q_i^l$, resulting in the updated question representation $q^l$ as

$$q^l = \sum_i P_i q_i^l. \tag{6}$$

The updated question representation $q^l$ after weighted pooling serves as the question input to the next reasoning or answering layer.

### 3.4 Answering Layer

Following [19, 2], we model VQA as a classification problem with pre-defined classes. Given the updated question representation at last reasoning layer $q^L$, a softmax layer is employed to classify $q^L$ into one of the possible answers as

$$p_{ans} = softmax(W_{ans} * q^L + b_{ans}). \tag{7}$$

Note instead of the softmax layer for predicting the correct answer, it is also possible to utilize LSTM or GRU decoder, taking $q^L$ as input, to generate free-form answers.

## 4   Experiments

### 4.1   Datasets and Evaluation Metrics

We conduct experiments on COCO-QA [19] and VQA [2]. The COCO-QA dataset is based on Microsoft COCO image data [13]. There are 78,736 training questions and 38,948 test ones, based on a total of 123,287 images. Four types of questions are provided, including *Object*, *Number*, *Color* and *Location*. Each type takes 70%, 7%, 17% and 6% of the whole dataset respectively.

In the VQA dataset, each image from the COCO data is annotated by Amazon Mechanical Turk (AMT) with three questions. It is the largest for VQA benchmark so far. There are 248,349, 121,512 and 244,302 questions for training, validation and testing, respectively. For each question, ten answers are provided to take consensus of annotators. Following [2], we choose the top 1,000 most frequent answers as candidate outputs, which constitutes 82.67% of the train+val answers.

Since we formulate VQA as a classification problem, mean classification accuracy is used to evaluate the model on the COCO-QA dataset. Besides, Wu-Palmer similarity (WUPS) [25] measure is also reported on COCO-QA dataset. WUPS calculates similarity between two words based on their longest common subsequence in the taxonomy tree. Following [19], we use thresholds 0.9 and 0.0 in our evaluation. VQA dataset provides a different kind of evaluation metric. Since ten ground truth answers are given, a predicted answer is considered to be correct when three or more ground truth answers match it. Otherwise, partial score is given.

### 4.2   Implementation Details

We implement our network using the public Torch computing framework. Before training, all question sentences are normalized to lower case where question marks are removed. These words are fed into GRU one by one. The whole answer with one or more words is regarded as a separate class. For extracting image features, each candidate region is cropped and resized to $224 \times 224$ before feeding into CNN.

For the COCO-QA dataset, we set the dimension of common latent space to 1,024. Since VQA dataset is larger than COCO-QA, we double the dimension of common latent space to adapt the data and classes. On each reasoning layer, we use one single layer in MLP. We test up to two reasoning layers. No further improvement is observed when using three or more layers.

| Methods | ACC. | Object | Number | Color | Location |
|---|---|---|---|---|---|
| Mean Pooling | 58.15 | 60.61 | 45.34 | 55.37 | 52.74 |
| Max Pooling | 59.37 | 62.11 | 45.70 | 55.91 | 53.63 |
| W/O Global | 60.87 | 63.32 | **46.68** | 58.66 | 55.49 |
| W/O Coord | 61.33 | 63.76 | 46.24 | 59.35 | 56.66 |
| Full Model | **61.99** | **64.53** | **46.68** | **59.81** | **56.82** |

Table 1: Comparison of ablation models. Models are trained and tested on COCO-QA [19] with one reasoning layer.

| Methods | ACC. | Object | Number | Color | Location | WUPS 0.9 | WUPS 0.0 |
|---|---|---|---|---|---|---|---|
| IMG+BOW [19] | 55.92 | 58.66 | 44.10 | 51.96 | 49.39 | 66.78 | 88.99 |
| 2VIS+BLSTM [19] | 55.09 | 58.17 | 44.79 | 49.53 | 47.34 | 65.34 | 88.64 |
| Ensemble [19] | 57.84 | 61.08 | 47.66 | 51.48 | 50.28 | 67.90 | 89.52 |
| ABC-CNN [3] | 58.10 | 62.46 | 45.70 | 46.81 | 53.67 | 68.44 | 89.85 |
| DPPnet [17] | 61.19 | - | - | - | - | 70.84 | 90.61 |
| SAN [29] | 61.60 | 64.50 | **48.60** | 57.90 | 54.00 | 71.60 | 90.90 |
| QRU (1) | 61.99 | 64.53 | 46.68 | 59.81 | 56.82 | 71.83 | 91.11 |
| QRU (2) | **62.50** | **65.06** | 46.90 | **60.50** | **56.99** | **72.58** | **91.62** |

Table 2: Evaluation results on COCO-QA dataset [19]. "QRU (1)" and "QRU (2)" refer to 1 and 2 reasoning layers incorporated in the system.

The network is trained in an end-to-end fashion using stochastic gradient descent with mini-batches of 100 samples and momentum 0.9. The learning rate starts from $10^{-3}$ and decreases by a factor of 10 when validation accuracy stops improving. We use dropout and gradient clipping to regularize the training process. Our model is denoted as QRU in following experiments.

## 4.3 Ablation Results

We conduct experiments to exam the usefulness of each component in our model. Specifically, we compare different question representation pooling mechanisms, i.e., *mean pooling* and *max pooling*. We also train two controlled models devoid of global image feature and spatial coordinate, denoted as *W/O Global* and *W/O Coord*. Table 1 shows the results.

The performance of mean and max pooling models are substantially worse than the full model, which uses weighted pooling. This indicates that our model benefits from the attention mechanism by looking at several image regions rather than only one or all of them. A drop of 1.12% in accuracy is observed if the global image feature is not modeled, confirming that inclusion of the whole image is important for capturing the global information. Without modeling spatial coordinates also leads to a drop in accuracy. Notably, the greatest deterioration is on the question type of *Object*. This is because the *Object* type seeks information around the object like "What is next to the stop sign?". Spatial coordinates help our model reason spatial relationship among objects.

## 4.4 Comparison with State-of-the-art

We compare performance in Tables 2 and 3 with experimental results on COCO-QA and VQA respectively. Table 2 shows that our model with only one reasoning layer already outperforms state-of-the-art 2-layer stacked attention network (SAN) [29]. Two reasoning layers give the best performance. We also report the per-category accuracy to show the strength and weakness of our model in Table 2. Our best model outperforms SAN by 2.6% and 2.99% in the question types of *Color* and *Location* respectively, and by 0.56% in *Object*.

Our analysis is that the SAN model puts its attention on coarser regions obtained from the activation of last convolutional layer, which may include cluttered and noisy background. In contrast, our model only deals with selected object proposal regions, which have the good chance to be objects. When answering questions involving objects, our model gives reasonable results. For the question type *Number*, since an object proposal may contain several objects, our counting ability is weakened. In fact, the counting task is a complete computer vision problem on its own.

| Methods | Open-Ended (test-dev) | | | | test-std | Multiple-Choice (test-dev) | | | | test-std |
|---|---|---|---|---|---|---|---|---|---|---|
| | All | Y/N | Num | Other | All | All | Y/N | Num | Other | All |
| BOWIMG [2] | 52.64 | 75.77 | 33.67 | 37.37 | - | 58.97 | 75.59 | 34.35 | 50.33 | - |
| LSTMIMG [2] | 53.74 | 78.94 | 35.24 | 36.42 | 54.06 | 57.17 | 78.95 | 35.80 | 43.41 | 57.57 |
| iBOWIMG [30] | 55.72 | 76.55 | 35.03 | 42.62 | 55.89 | 61.68 | 76.68 | 37.05 | 54.44 | 61.97 |
| DPPnet [17] | 57.22 | 80.71 | **37.24** | 41.71 | 57.36 | 62.48 | 80.79 | 38.94 | 52.16 | 62.69 |
| SAN [29] | 58.70 | 79.30 | 36.60 | 46.10 | 58.90 | - | - | - | - | - |
| WR Sel [20] | - | - | - | - | - | 62.44 | 77.62 | 34.28 | 55.84 | 62.43 |
| FDA [9] | 59.24 | 81.14 | 36.16 | 45.77 | 59.54 | 64.01 | 81.50 | **39.00** | 54.72 | 64.18 |
| DMN+ [26] | 60.37 | 80.75 | 37.00 | **48.25** | 60.36 | - | - | - | - | - |
| QRU (1) | 59.26 | 80.98 | 35.93 | 45.99 | 59.44 | 63.96 | 81.00 | 37.08 | 55.48 | 64.13 |
| QRU (2) | **60.72** | **82.29** | 37.02 | 47.67 | **60.76** | **65.43** | 82.24 | 38.69 | **57.12** | **65.43** |

Table 3: Evaluation results on VQA dataset [2]. "QRU (1)" and "QRU (2)" refer to 1 and 2 reasoning layers incorporated in the system.



| | |
|---|---|
| Original | What next to two other open laptops? |
| Before updating | What next to each other dipicting smartphones? <br> What next to two boys? <br> What hooked up to two computers? <br> What next to each other with visible piping? <br> What next to two pair of shoes? |
| After updating with one reasoning layer | What are there laying down with two remotes? <br> What next to each other depicting smartphones? <br> What hooked up to two computers? <br> What next to each other with monitors? <br> What cubicle with four differnet types of computers? |
| After updating with two reasoning layers | What plugged with wires? <br> What next to each other with monitors? <br> What are open at the table with cell phones? <br> What is next to the monitor? <br> What sits on the desk along with 2 monitors? |

Figure 3: Retrieved questions before and after update from COCO-QA dataset [19].

Table 3 shows that our model yields prominent improvement on the *Other* type when compared with other models [2, 30, 17] that use global representation of the image. Object proposals in our model are useful since the *Other* type contains questions such as "What color ⋯", "What kind ⋯", "Where is ⋯", etc. Our model outperforms that of [20] by 3% where the latter also exploits object proposals. Compared with [20], we use less number of object proposals, demonstrating the effectiveness of our approach. This table also reveals that our model with two reasoning layers achieve state-of-the-art results for both open-ended and multiple-choice tasks.

### 4.5 Qualitative Analysis

To understand the ability of our model in updating question representation, we show an image and several questions in Figure 3. The retrieved questions from the test set are based on the cosine similarities to the original question before and after our model updates the representation. It is notable that before update, 4 out of the top 5 similar questions begin with "What next". This is because GRU acts as the language model, making the obtained questions share similar language structure. After we update question representation, the resulting ones are more related to image content regarding objects *computers* and *monitors* while the originally retrieved questions contain irrelevant words like *boys* and *shoes*. The retrieved questions become even more informative using two reasoning layers.

We visualize a few attention masks generated by our model in Figure 4. Visualization is created by soft masking the image with a mask created by summing weights of each region. The mask is normalized with maximum value 1 followed by small Gaussian blur. Our model is capable of putting attention on important regions closely relevant to the question. To answer the question "What is the color of the snowboard?", the proposed model finds the snowboard. For the other question "The man holding what on top of a snow covered hill?", it is required to infer the relation among person, snow covered hill, and snowboard. With these attention masks, it is possible to predict correct answers since irrelevant image regions are ruled out. More examples are shown in Figure 5.

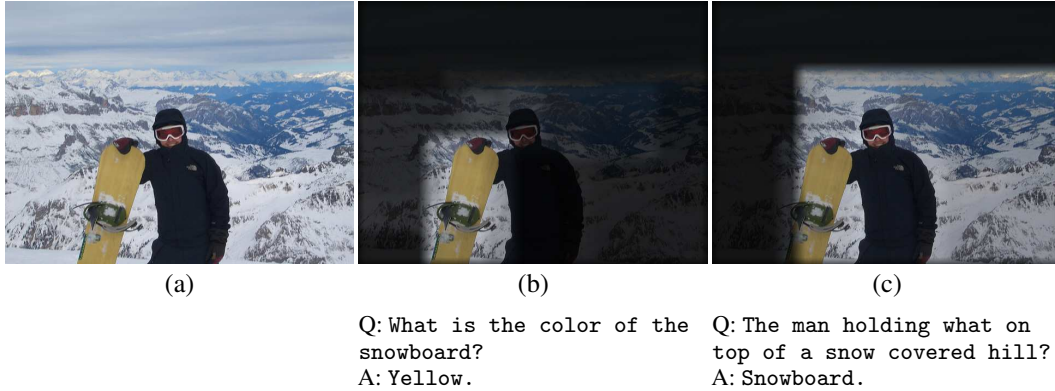

| (a) | (b) | (c) |
|---|---|---|

Q: What is the color of the
snowboard?
A: Yellow.

Q: The man holding what on
top of a snow covered hill?
A: Snowboard.

Figure 4: Visualization of attention masks. Our model learns to attend particular image regions that are relevant to the question.

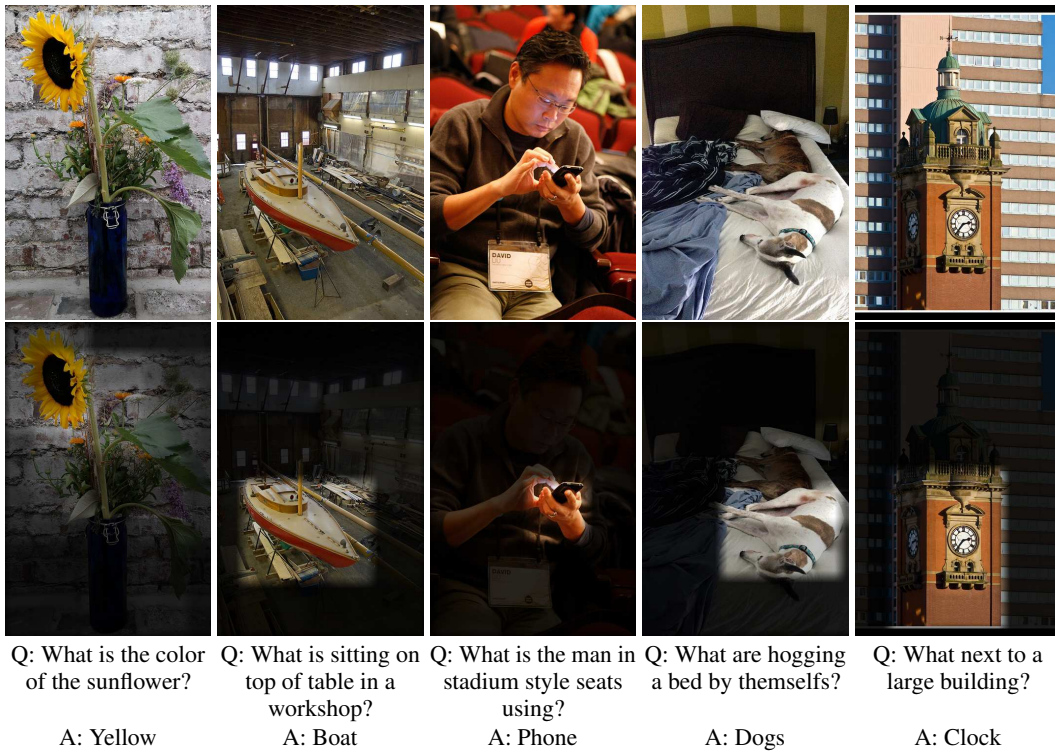

Q: What is the color of the sunflower?

A: Yellow

Q: What is sitting on top of table in a workshop?

A: Boat

Q: What is the man in stadium style seats using?

A: Phone

Q: What are hogging a bed by themselfs?

A: Dogs

Q: What next to a large building?

A: Clock

Figure 5: Visualization of more attention masks.

## 5 Conclusion

We have proposed an end-to-end trainable neural network for VQA. Our model learns to answer questions by updating question representation and inferring over a set of image regions with multilayer perceptron. Visualization of attention masks demonstrates the ability of our model to focus on image regions highly related to questions. Experimental results are satisfying on the two challenging VQA datasets. Future work includes improving object counting ability and word-region relation.

## Acknowledgements

This work is supported by a grant from the Research Grants Council of the Hong Kong SAR (project No. 2150760) and by the National Science Foundation China, under Grant 61133009. We thank NVIDIA for providing Ruiyu Li a Tesla K40 GPU accelerator for this work.

# References

[1] J. Andreas, M. Rohrbach, T. Darrell, and D. Klein. Learning to compose neural networks for question answering. *arXiv preprint arXiv:1601.01705*, 2016.

[2] S. Antol, A. Agrawal, J. Lu, M. Mitchell, D. Batra, C. Lawrence Zitnick, and D. Parikh. Vqa: Visual question answering. In *ICCV*, pages 2425–2433, 2015.

[3] K. Chen, J. Wang, L.-C. Chen, H. Gao, W. Xu, and R. Nevatia. Abc-cnn: An attention based convolutional neural network for visual question answering. *arXiv preprint arXiv:1511.05960*, 2015.

[4] J. Deng, W. Dong, R. Socher, L.-J. Li, K. Li, and L. Fei-Fei. Imagenet: A large-scale hierarchical image database. In *CVPR*, pages 248–255, 2009.

[5] H. Fang, S. Gupta, F. Iandola, R. K. Srivastava, L. Deng, P. Dollár, J. Gao, X. He, M. Mitchell, J. C. Platt, et al. From captions to visual concepts and back. In *CVPR*, pages 1473–1482, 2015.

[6] R. Girshick, J. Donahue, T. Darrell, and J. Malik. Rich feature hierarchies for accurate object detection and semantic segmentation. In *CVPR*, pages 580–587, 2014.

[7] B. Hu, Z. Lu, H. Li, and Q. Chen. Convolutional neural network architectures for matching natural language sentences. In *NIPS*, pages 2042–2050, 2014.

[8] R. Hu, H. Xu, M. Rohrbach, J. Feng, K. Saenko, and T. Darrell. Natural language object retrieval. *arXiv preprint arXiv:1511.04164*, 2015.

[9] I. Ilija, Y. Shuicheng, and F. Jiashi. A focused dynamic attention model for visual question answering. *arXiv preprint arXiv:1604.01485*, 2016.

[10] A. Karpathy and L. Fei-Fei. Deep visual-semantic alignments for generating image descriptions. In *CVPR*, pages 3128–3137, 2015.

[11] R. Kiros, Y. Zhu, R. R. Salakhutdinov, R. Zemel, R. Urtasun, A. Torralba, and S. Fidler. Skip-thought vectors. In *NIPS*, pages 3276–3284, 2015.

[12] A. Krizhevsky, I. Sutskever, and G. E. Hinton. Imagenet classification with deep convolutional neural networks. In *NIPS*, pages 1097–1105, 2012.

[13] T.-Y. Lin, M. Maire, S. Belongie, J. Hays, P. Perona, D. Ramanan, P. Dollár, and C. L. Zitnick. Microsoft coco: Common objects in context. In *ECCV*, pages 740–755, 2014.

[14] L. Ma, Z. Lu, and H. Li. Learning to answer questions from image using convolutional neural network. *arXiv preprint arXiv:1506.00333*, 2015.

[15] M. Malinowski and M. Fritz. A multi-world approach to question answering about real-world scenes based on uncertain input. In *NIPS*, pages 1682–1690, 2014.

[16] M. Malinowski, M. Rohrbach, and M. Fritz. Ask your neurons: A neural-based approach to answering questions about images. In *ICCV*, pages 1–9, 2015.

[17] H. Noh, P. H. Seo, and B. Han. Image question answering using convolutional neural network with dynamic parameter prediction. *arXiv preprint arXiv:1511.05756*, 2015.

[18] B. Peng, Z. Lu, H. Li, and K.-F. Wong. Towards neural network-based reasoning. *arXiv preprint arXiv:1508.05508*, 2015.

[19] M. Ren, R. Kiros, and R. Zemel. Exploring models and data for image question answering. In *NIPS*, pages 2935–2943, 2015.

[20] K. J. Shih, S. Singh, and D. Hoiem. Where to look: Focus regions for visual question answering. *arXiv preprint arXiv:1511.07394*, 2015.

[21] K. Simonyan and A. Zisserman. Very deep convolutional networks for large-scale image recognition. *arXiv preprint arXiv:1409.1556*, 2014.

[22] S. Sukhbaatar, A. Szlam, J. Weston, and R. Fergus. Weakly supervised memory networks. *arXiv preprint arXiv:1503.08895*, 2015.

[23] I. Sutskever, O. Vinyals, and Q. V. Le. Sequence to sequence learning with neural networks. *arXiv preprint arXiv:1409.3215*, 2014.

[24] O. Vinyals, A. Toshev, S. Bengio, and D. Erhan. Show and tell: A neural image caption generator. In *CVPR*, pages 3156–3164, 2015.

[25] Z. Wu and M. Palmer. Verbs semantics and lexical selection. In *ACL*, pages 133–138, 1994.

[26] C. Xiong, S. Merity, and R. Socher. Dynamic memory networks for visual and textual question answering. *arXiv preprint arXiv:1603.01417*, 2016.

[27] H. Xu and K. Saenko. Ask, attend and answer: Exploring question-guided spatial attention for visual question answering. *arXiv preprint arXiv:1511.05234*, 2015.

[28] K. Xu, J. Ba, R. Kiros, A. Courville, R. Salakhutdinov, R. Zemel, and Y. Bengio. Show, attend and tell: Neural image caption generation with visual attention. *arXiv preprint arXiv:1502.03044*, 2015.

[29] Z. Yang, X. He, J. Gao, L. Deng, and A. Smola. Stacked attention networks for image question answering. *arXiv preprint arXiv:1511.02274*, 2015.

[30] B. Zhou, Y. Tian, S. Sukhbaatar, A. Szlam, and R. Fergus. Simple baseline for visual question answering. *arXiv preprint arXiv:1512.02167*, 2015.

[31] C. L. Zitnick and P. Dollár. Edge boxes: Locating object proposals from edges. In *ECCV*, pages 391–405, 2014.

